# Fast and Robust Classification using Asymmetric AdaBoost and a Detector Cascade

**Paul Viola and Michael Jones**
Mistubishi Electric Research Lab
Cambridge, MA
*viola@merl.com* and *mjones@merl.com*

## Abstract

This paper develops a new approach for extremely fast detection in domains where the distribution of positive and negative examples is highly skewed (e.g. face detection or database retrieval). In such domains a cascade of simple classifiers each trained to achieve high detection rates and modest false positive rates can yield a final detector with many desirable features: including high detection rates, very low false positive rates, and fast performance. Achieving extremely high detection rates, rather than low error, is not a task typically addressed by machine learning algorithms. We propose a new variant of AdaBoost as a mechanism for training the simple classifiers used in the cascade. Experimental results in the domain of face detection show the training algorithm yields significant improvements in performance over conventional AdaBoost. The final face detection system can process 15 frames per second, achieves over 90% detection, and a false positive rate of 1 in a 1,000,000.

## 1 Introduction

In many applications fast classification is almost as important as accurate classification. Common examples include robotics, user interfaces, and classification in large databases. In this paper we demonstrate our approach in the domain of low latency, sometimes called "real-time", face detection. An extremely fast face detector is a critical component in many applications. User-interfaces can be constructed which detect the presence and number of users. Teleconference systems can automatically devote additional bandwidth to participant's faces. Video security systems can record facial images of individuals after unauthorized entry.

Recently we presented a real-time face detection system which scans video images at 15 frames per second [8] yet achieves detection rates comparable with the best published results (e.g. [7]) [1] Face detection is a scanning process, in which a face classifier is evaluated at every scale and location within each image. Since there are about 50,000 unique scales

and locations in a typical image, this amounts to evaluating the face classifier 750,000 times per second.

One key contribution of our previous work was the introduction of a classifier cascade. Each stage in this cascade was trained using AdaBoost until the required detection performance was achieved [2]. In this paper we present a new training algorithm designed specifically for a classifier cascade called asymmetric AdaBoost. The algorithm is a generalization of that given in Singer and Shapire [6]. Many of the formal guarantees presented by Singer and Shapire also hold for this new algorithm. The paper concludes with a set of experiments in the domain of face detection demonstrating that asymmetric AdaBoost yields a significant improvement in detection performance over conventional boosting.

## 2 Classifier Cascade

In the machine learning community it is well known that more complex classification functions yield lower training errors yet run the risk of poor generalization. If the main consideration is test set error, structural risk minimization provides a formal mechanism for selecting a classifier with the right balance of complexity and training error [1].

Another significant consideration in classifier design is computational complexity. Since time and error are fundamentally different quantities, no theory can simply select the optimal trade-off. Nevertheless, for many classification functions computation time is directly related to the structural complexity. In this way temporal risk minimization is clearly related to structural risk minimization.

This direct analogy breaks down in domains where the distribution over the class labels is highly skewed. For example, in the domain of face detection, there are at most a few dozen faces among the 50,000 sub-windows in an image. Surprisingly in these domains it is often possible to have the best of both worlds: high detection rates and extremely fast classification. The key insight is that while it may be impossible to construct a simple classifier which can achieve a low training/test error, in some cases it is possible to construct a simple classifier with a very low false negative rate. For example, in the domain of face detection, we have constructed an extremely fast classifier with a very low false negative rate (i.e. it almost never misses a face) and a 50% false positive rate. Such a detector might be more accurately called a classification pre-filter: when an image region is labeled 'non-face' then it can be immediately discarded, but when a region is labeled 'face' then further classification effort is required. Such a pre-filter can be used as the first stage in a cascade of classifiers (see Figure 1).

In our face detection application (described in more detail in Section 5) the cascade has 38 stages. Even though there are many stages, most are not evaluated for a typical non-face input window since the early stages weed out many non-faces. In fact, over a large test set, the average number of stages evaluated is less than 2. In a cascade, computation time and detection rate of the first few stages is critically important to overall performance. The remainder of the paper describes techniques for training cascade classifiers which are efficient yet effective.

## 3 Using Boosting to Train the Cascade

In general almost any form of classifier can be used to construct a cascade; the key properties are that computation time and the detection rate can be adjusted. Examples include support vector machines, perceptrons, and nearest neighbor classifiers. In the case of an SVM computation time is directly related to the number of support vectors and detection rate is related to the margin threshold [1].

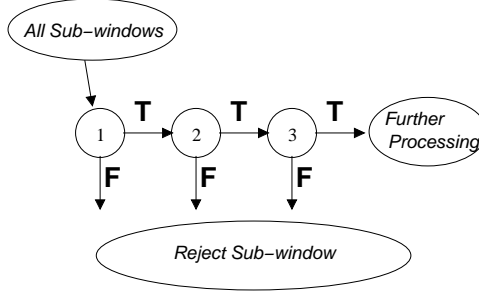

Figure 1: Schematic depiction of a detection cascade. A sequence of classifiers are applied to every example. The initial classifier eliminates a large number of negative examples with very little processing. Subsequent stages eliminate additional negatives but require additional computation. Extremely few negative examples remain after several stages.

In our system each classifier in the cascade is a single layer perceptron whose input is a set of computationally efficient binary features. The computational cost of each classifier is then simply the number of input features. The detection rate is adjusted by changing the threshold (or bias).

Much of the power of our face detection system comes from the very large and varied set of features available. In our experiments over 6,000,000 different binary features were available for inclusion in the final classifiers (see Figure 4 for some example features). The efficiency of each classifier, and hence the efficiency of the cascade, is ensured because a very small number of features are included in the early stages; the first stage has 1 (!) feature, the second stage 5 features, then 20, and then 50. See Section 5 for a brief description of the feature set. The main contribution of this paper is the adaptation of AdaBoost for the task of feature selection and classifier learning.

Though it is not widely appreciated, AdaBoost provides a principled and highly efficient mechanism for feature selection[2, 6]. If the set of weak classifiers is simply the set of binary features (this is often called boosting stumps) each round of boosting adds a single feature to the set of current features.

AdaBoost is an iterative process in which each round selects a weak classifier, $h_t()$, which minimizes:

$$Z_t = \sum_i D_t(i) exp\left(-y_i h_t(x_i)\right). \tag{1}$$

Following the notation of Shapire and Singer, $D_t(i)$ is the weight on example $i$ at round $t$, $y_i \in \{-1, 1\}$ is the target label of the example, $x_i$ is the example, and $h_t()$ is a confidence rated binary classifier[6]. After every round the weights are updated as follows:

$$D_{t+1}(i) = \frac{D_t(i) exp\left(-y_i h_t(x_i)\right)}{Z_t}. \tag{2}$$

The classifier $h_t()$ takes on two possible values $h_+ = \log \sqrt{\frac{W_{++}}{W_{+-}}}$ and $h_- = \log \sqrt{\frac{W_{--}}{W_{-+}}}$, where $W_{pq}$ is the weight of the examples given the label $p$ which have true label $q$. These predictions insure that the weights on the next round are *balanced*: that the relative weights of positive and negative examples one each side of the classification boundary are equal.

Minimizing $Z_t$ minimizes the weighted *exponential loss* at round $t$. Minimizing $Z_t$ in each round is also a greedy technique for minimizing $\prod_t Z_t$ which is an upper bound on the training error of the strong classifier. It has also been observed that the example weights

are directly related to example margin, which leads to a principled argument for AdaBoost's generalization capabilities [5].

The key advantage of AdaBoost as a feature selection mechanism, over competitors such as the wrapper method [3], is the speed of learning. Given the constraint that the search over features is greedy, AdaBoost efficiently selects the feature which minimizes $\prod_t Z_t$, a surrogate for overall classification error. The entire dependence on previously selected features is efficiently and compactly encoded using the example weights. As a result, the addition of the 100th feature requires no more effort than the selection of the first feature. [2]

## 4  Asymmetric AdaBoost

One limitation of AdaBoost arises in the context of skewed example distributions and cascaded classifiers: AdaBoost minimizes a quantity related to classification error; it does not minimize the number of false negatives. Given that the final form of the classifier is a weighted majority of features, the detection and false positive rates are adjustable after training. Unfortunately feature selection proceeds as if classification error were the only goal, and the features selected are not optimal for the task of rejecting negative examples.

One naive scheme for "fixing" AdaBoost is to modify the initial distribution over the training examples. If we hope to minimize false negatives then the weight on positive examples could be increased so that the minimum error criteria will also have very few false negatives. We can formalize this intuitive approach as follows. Recall that AdaBoost is a scheme which minimizes:

$$\prod_t Z_t = \sum_i exp\left(-y_i \sum_t h_t(x_i)\right) \tag{3}$$

Each term in the summation is bounded above by a simple *loss* function:

$$exp\left(-y_i \sum_t h_t(x_i)\right) \geq Loss(i) = \begin{cases} 1 & \text{if } y_i \neq C(x_i), \\ 0 & \text{otherwise} \end{cases} \tag{4}$$

where $C(x_i)$ is the class assigned by the boosted classifier. As a result, minimizing $\prod_t Z_t$ minimizes an upper bound on simple loss.

We can introduce a related notion of asymmetric loss:

$$ALoss(i) = \begin{cases} \sqrt{k} & \text{if } y_i = 1 \text{ and } C(x_i) = -1, \\ \frac{1}{\sqrt{k}} & \text{if } y_i = -1 \text{ and } C(x_i) = 1, \\ 0 & \text{otherwise.} \end{cases} \tag{5}$$

where false negatives cost $k$ times more than false positives. Note that $ALoss(i) = exp\left(y_i \log \sqrt{k}\right) Loss(i)$. If we take the bound in Equation 4 and multiply both sides by $exp(y_i \sqrt{k})$ we obtain a bound on the asymmetric loss: $exp\left(-y_i \sum_t h_t(x_i)\right) exp\left(y_i \sqrt{k}\right) \geq ALoss(i)$.

Minimization of this bound can be achieved using AdaBoost by pre-weighting each example by $exp\left(y_i \log \sqrt{k}\right)$. The derivation is identical to that of Equation 3. Expanding

Equation 2 repeatedly for $D_t(i)$ in terms of $D_{t-1}(i)$ we arrive at,

$$D_{t+1}(i) = \frac{exp\left(-y_i \sum_t h_t(x_i)\right) exp\left(y_i \log \sqrt{k}\right)}{\prod_t Z_t}, \tag{6}$$

where the second term in the numerator arises because of the initial asymmetric weighting. Noticing that the left hand side must sum to 1 yields the following equality,

$$\prod_t Z_t = \sum_i \left( exp\left(-y_i \sum_t h_t(x_i)\right) exp\left(y_i \log \sqrt{k}\right)\right). \tag{7}$$

Therefore AdaBoost minimizes the required bound on asymmetric loss.

Unfortunately this naive technique is only somewhat effective. The main reason is AdaBoost's balanced reweighting scheme. As a result the initially asymmetric example weights are immediately lost. Essentially the AdaBoost process is too greedy. The first classifier selected absorbs the entire effect of the initial asymmetric weights. The remaining rounds are entirely symmetric.

We propose a closely related approach that results in the minimization of the same bound, which nevertheless preserves the asymmetric loss throughout all rounds. Instead of applying the necessary asymmetric multiplier $exp\left(y_i \log \sqrt{k}\right)$ at the first round of an $N$ round process, the nth root $exp\left(\frac{1}{N} y_i \log \sqrt{k}\right)$ is applied before each round. Referring to Equation 6 we can see the final effect is the same; this preserves the bound on asymmetric loss. But the effect on the training process is quite different. In order to demonstrate this approach we generated an artificial data set and learned strong classifiers containing 4 weak classifiers. The results are shown inFigure 2. In this figure we can see that all but the first weak classifier learned by the naive rule are poor, since they each balance positive and negative errors. The final combination of these classifiers cannot yield high detection rates without introducing many false positives. All the weak classifiers generated by the proposed Asymmetric Adaboost rule are consistent with asymmetric loss and the final strong classifier yields very high detection rates and modest false positive rates.

One simple reinterpretation of this distributed scheme for asymmetric reweighting is as a reduction in the positive confidence of each weak classifier $h_t'() = h_t() - \frac{1}{N} \log \sqrt{k}$. This forces each subsequent weak classifier to focus asymmetrically on postive examples.

## 5  Experiments

We performed two experiments in the domain of frontal face detection to demonstrate the advantages of asymmetric AdaBoost. Experiments follow the general form, though differ in details, from those presented in Viola and Jones [8]. In each round of boosting one of a very large set of binary features are selected. These features, which we call rectangle features, are briefly described in Figure 4.

In the first experiment a training and test set containing faces and non-faces of a fixed size were acquired (faces were scaled to a size $24 \times 24$ pixels). The training set consisted of 1500 face examples and 5000 non-face examples. Test data included 900 faces and 5000 non-faces. The face examples were manually cropped from a large collection of Web images while the non-face examples were randomly chosen patches from Web images that were known not to contain any faces.

Naive asymetric AdaBoost and three parameterizations of Asymmetric AdaBoost were used to train classifiers with 4 features on this data. Figure 3 shows the ROC curves on

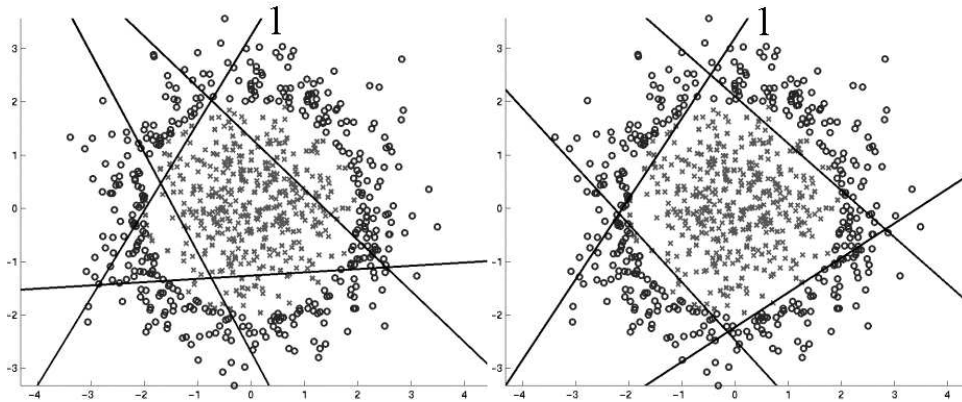

Figure 2: Two simple examples: positive examples are 'x', negative 'o' and weak classifiers are linear separators. On the left is the naive asymetric result. The first feature selected is labelled '1'. Subsequent features attempt to balance positive and negative errors. Notice that no linear combination of the 4 weak classifiers can achieve a low false positive and low false negative rate. On the right is the asymetric boosting result. After learning 4 weak classifier the positives are well modelled and most of the negative are rejected.

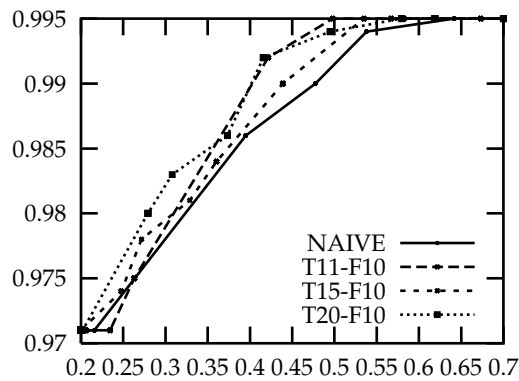

Figure 3: ROC curves for four boosted classifier with 4 features. The first is naive asymmetric boosting. The other three results are for the new asymmetric approach, each using slightly different parameters. The ROC curve has been cropped to show only the region of interest in training a cascaded detector, the high detection rate regime. Notice that that at 99% detection asymmetric Adaboost cuts the false positive by about 20%. This will significantly reduce the work done by later stages in the cascade.

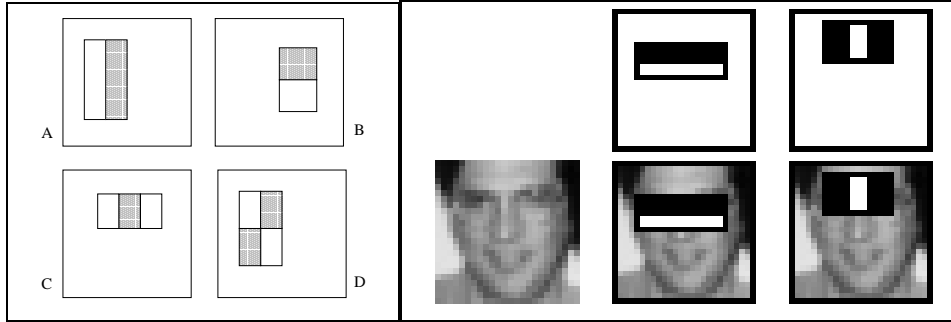

Figure 4: **Left:** Example rectangle features shown relative to the enclosing detection window. The sum of the pixels which lie within the white rectangles are subtracted from the sum of pixels in the gray rectangles. A threshold operation is then applied to yield a binary output. Two-rectangle features are shown in (A) and (B). Figure (C) shows a three-rectangle feature, and (D) a four-rectangle feature. **Right:** The first two example feature selected by the boosting process. Notice that the first feature relies on the fact that the horizontal region of the eyes is darker than the horizontal region of the cheeks. The second feature, whose selection is conditioned on the first, acts to distinguish horizontal edges from faces by looking for a strong vertical edge near the nose.

test data for the three classifiers. The key result here is that at high detection rates the false positive rate can be reduced significantly.

In the second experiment, naive and asymmetric AdaBoost were used to train two different complete cascaded face detectors. Performance of each cascade was determined on a real-world face detection task, which requires scanning of the cascade across a set of large images which contain embedded faces.

The cascade training process is complex, and as a result comparing detection results is useful but potentially risky. While the data used to train the two cascades were identical, the performance of earlier stages effects the selection of non-faces used to train later stages. As a result different non-face examples are used to train the corresponding stages for the Naive and Asymmetric results.

Layers were added to each of the cascades until the number of false positives was reduced below 100 on a validation set. For normal boosting this occurred with 34 layers. For asymmetric AdaBoost this occurred with 38 layers. Figure 5 shows the ROC curves for the resulting face detectors on the MIT+CMU [4] test set. [3] Careful examination of the ROC curves show that the asymmetric cascade reduces the number of false positives significantly. At a detection rate of 91% the reduction is by a factor of 2.

## 6   Conclusions

We have demonstrated that a cascade classification framework can be used to achieve fast classification, high detection rates, and very low false positive rates. The goal for each classifier in the cascade is not low error, but instead extremely high detection rates and modest false positive rates. If this is achieved, each classifier stage can be used to filter out and discard many negatives.

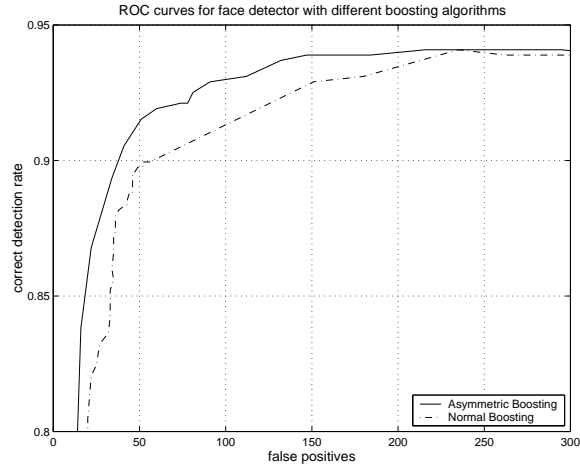

Figure 5: ROC curves comparing the accuracy of two full face detectors, one trained using normal boosting and the other with asymmetric AdaBoost. Again, the detector trained using asymmetric AdaBoost is more accurate over a wide range of false positive values.

Many modern approaches for classification focus entirely on the minimization of errors. Questions of relative loss only arise in the final tuning of the classifier. We propose a new training algorithm called asymmetric AdaBoost which performs learning and efficient feature selection with the fundamental goal of achieving high detection rates. Asymmetric AdaBoost is a simple modification of the "confidence-rated" boosting approach of Singer and Shapire. Many of their derivations apply to this new approach as well.

Experiments have demonstrated that asymmetric AdaBoost can lead to significant improvements both in classification speed and in detection rates.

## Footnotes

[1] In order to achieve real-time speeds other systems often resort to skin color filtering in color images or motion filtering in video images. These simple queues are useful but unreliable. In large image databases color and motion are often unavailable. Our system detects faces using only static monochrome information.

[2]Given that there are millions of features and thousands of examples, the boosting process requires days of computation. Many other techniques while feasible for smaller problems are likely to be infeasible for this sort of problem.

[3]Note: the detection and false positive rates for the simple 40 feature experiment and the more complex cascaded experiment are *not directly comparable*, since the test sets are quite different.

## References

[1] Corinna Cortes and Vladimir Vapnik. Support-vector networks. *Machine Learning*, 20, 1995.

[2] Yoav Freund and Robert E. Schapire. A decision-theoretic generalization of on-line learning and an application to boosting. In *Computational Learning Theory: Eurocolt '95*, pages 23–37. Springer-Verlag, 1995.

[3] G. John, R. Kohavi, and K. Pfleger. Irrelevant features and the subset selection problem. In *Machine Learning Conference*, pages 121–129. Morgan Kaufmann, 1994.

[4] H. Rowley, S. Baluja, and T. Kanade. Neural network-based face detection. In *IEEE Patt. Anal. Mach. Intell.*, volume 20, pages 22–38, 1998.

[5] R. E. Schapire, Y. Freund, P. Bartlett, and W. S. Lee. Boosting the margin: a new explanation for the effectiveness of voting methods. *Ann. Stat.*, 26(5):1651–1686, 1998.

[6] Robert E. Schapire and Yoram Singer. Improved boosting algorithms using confidence-rated predictions. *Machine Learning*, 37:297–336, 1999.

[7] H. Schneiderman and T. Kanade. A statistical method for 3D object detection applied to faces and cars. In *Computer Vision and Pattern Recognition*, 2000.

[8] Paul Viola and Michael J. Jones. Robust real-time object detection. In *Proc. of IEEE Workshop on Statistical and Computational Theories of Vision*, 2001.
